# Learning Higher-Order Graph Structure with Features by Structure Penalty

**Shilin Ding**[1]*, **Grace Wahba**[1,2,3]*, **and Xiaojin Zhu**[2]*
Department of {[1]Statistics, [2]Computer Sciences, [3]Biostatistics and Medical Informatics}
University of Wisconsin-Madison, WI 53705
{sding, wahba}@stat.wisc.edu, jerryzhu@cs.wisc.edu

## Abstract

In discrete undirected graphical models, the conditional independence of node labels $Y$ is specified by the graph structure. We study the case where there is another input random vector $X$ (e.g. observed features) such that the distribution $P(Y \mid X)$ is determined by functions of $X$ that characterize the (higher-order) interactions among the $Y$'s. The main contribution of this paper is to learn the graph structure and the functions conditioned on $X$ at the same time. We prove that discrete undirected graphical models with feature $X$ are equivalent to multivariate discrete models. The reparameterization of the potential functions in graphical models by conditional log odds ratios of the latter offers advantages in representation of the conditional independence structure. The functional spaces can be flexibly determined by kernels. Additionally, we impose a Structure Lasso (SLasso) penalty on groups of functions to learn the graph structure. These groups with overlaps are designed to enforce hierarchical function selection. In this way, we are able to shrink higher order interactions to obtain a sparse graph structure.

## 1 Introduction

In undirected graphical models (UGMs), a graph is defined as $G = (V, E)$, where $V = \{1, \cdots, K\}$ is the set of nodes and $E \subset V \times V$ is the set of edges between the nodes. The graph structure specifies the conditional independence among nodes. Much prior work has focused on graphical model structure learning without conditioning on $X$. For instance, Meinshausen and Bühlmann [1] and Peng *et al.* [2] studied sparse covariance estimation of Gaussian Markov Random Fields. The covariance matrix fully determines the dependence structure in the Gaussian distribution. But it is not the case for non-elliptical distributions, such as the discrete UGMs. Ravikumar *et al.* [3] and Höfling and Tibshirani [4] studied variable selection of Ising models based on $l_1$ penalty. Ising models are special cases of discrete UGMs with (usually) only pairwise interactions, and without features. We focused on discrete UGMs with both higher order interactions and features. It is important to note that the graph structure may change conditioned on different $X$'s, thus our approach may lead to better estimates and interpretation.

In addressing the problem of structure learning with features, Liu *et al.* [5] assumed Gaussian distributed $Y$ given $X$, and they partitioned the space of $X$ into bins. Schmidt *et al.* [6] proposed a framework to jointly learn pairwise CRFs and parameters with block-$l_1$ regularization. Bradley and Guestrin [7] learned tree CRF that recovers a max spanning tree of a complete graph based on heuristic pairwise link scores. These methods utilize only pairwise information to scale to large graphs. The closest work is Schmidt and Murphy [8], which examined the higher-order graphical structure

learning problem without considering features. They used an active set method to learn higher order interactions in a greedy manner. Their model is over-parameterized, and the hierarchical assumption is sufficient but not necessary for conditional independence in the graph.

To the best of our knowledge, no previous work addressed the issue of graph structure learning of all orders while conditioning on input features. Our contributions include a reparemeterization of UGMs with bivariate outcomes into multivariate Bernoulli (MVB) models. The set of conditional log odds ratios in MVB models are complete to represent the effects of features on responses and their interactions at all levels. The sparsity in the set of functions are sufficient and necessary for the conditional independence in the graph, i.e., two nodes are conditionally independent iff the pairwise interaction is constant zero; and the higher order interaction among a subset of nodes means none of the variables is separable from the others in the joint distribution.

To obtain a sparse graph structure, we impose Structure Lasso (SLasso) penalty on groups of functions with overlaps. SLasso can be viewed as group lasso with overlaps. Group lasso [9] leads to selection of variables in groups. Jacob *et al.* [10] considered the penalty on groups with arbitrary overlaps. Zhao *et al.* [11] set up the general framework for hierarchical variable selection with overlapping groups, which we adopt here for the functions. Our groups are designed to shrink higher order interactions similar to hierarchical inclusion restriction in Schimdt and Murphy [8]. We give a proximal linearization algorithm that efficiently learns the complete model. Global convergence is guaranteed [12]. We then propose a greedy search algorithm to scale our method up to large graphs as the number of parameters grows exponentially.

## 2 Conditional Independence in Discrete Undirected Graphical Models

In this section, we first discuss the relationship between the multivariate Bernoulli (MVB) model and the UGM whose nodes are binary, i.e. $Y_i = 0$ or $1$. At the end, we will give the representation of the general discrete UGM where $Y_i$ takes value in $\{0, \cdots, m-1\}$. In UGMs, the distribution of multivariate discrete random variables $Y_1, \ldots, Y_K$ given $X$ is:

$$P(Y_1 = y_1, \ldots, Y_K = y_K | X) = \frac{1}{Z(X)} \prod_{C \in \mathcal{C}} \Phi_C(y_C; X) \tag{1}$$

where $Z(X)$ is the normalization factor. The distribution is factorized according to the cliques in the graph. A clique $C \subseteq \Omega = \{1, \ldots, K\}$ is the set of nodes that are fully connected. $\Phi_C(y_C; X)$ is the potential function on $C$, indexed by $y_C = (y_i)_{i \in C}$. This factorization follows from the Markov property: any two nodes not in a clique are conditionally independent given others [13]. So $\mathcal{C}$ does not have to comply with the graph structure, as long as it is sufficient. For example, the most general choice for any given graph is $\mathcal{C} = \{\Omega\}$. See Theorem 2.1 and Example 2.1 for details.

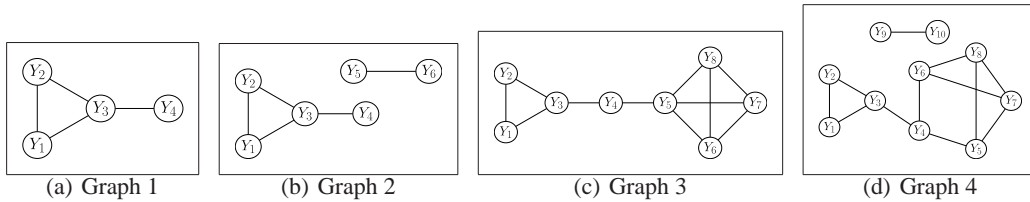

(a) Graph 1        (b) Graph 2        (c) Graph 3        (d) Graph 4

Figure 1: Graphical model examples.

Given the graph structure, the potential functions characterize the distribution on the graph. But if the graph is unknown in advance, estimating the potential functions on all possible cliques tends to be over-parameterized [8]. Furthermore, $\log \Phi_C(y_C; X) = 0$ is sufficient for the conditional independence among the nodes but not necessary (see Example 2.1). To avoid these problems, we introduce the MVB model that is equivalent to (1) with binary nodes, i.e. $Y_i = 0$ or $1$. The MVB distribution is:

$$P(Y_1 = y_1, \ldots, Y_K = y_k | X = x) = \exp\Big\{ \sum_{\omega \in \Psi_K} y^\omega f^\omega - b(f) \Big\} \tag{2}$$

$$= \exp\Big\{ y_1 f^1(x) + \cdots + y_K f^K(x) + \cdots + y_1 y_2 f^{1,2}(x) + \cdots + y_1 \ldots y_K f^{1,\ldots,K}(x) - b(f) \Big\}$$

Here, we use the following notations. Let $\overline{\Psi}_K$ be the power set of $\Omega = \{1, \ldots, K\}$, and use $\Psi_K = \overline{\Psi}_K - \{\emptyset\}$ to index the $2^K - 1$ $f^\omega$'s in (2). Let $\omega$ denotes a set in $\Psi_K$, define $\mathcal{Y} = (y^1, \cdots, y^\omega, \cdots, y^\Omega)$ be the augmented response with $y^\omega = \prod_{i \in \omega} y_i$. And $f = (f^1, \ldots, f^\omega, \ldots, f^\Omega)$ is the vector of conditional log odds ratios [14]. We assume $f^\omega$ is in a Reproducing Kernel Hilbert Space (RKHS) $\mathcal{H}^\omega$ with kernel $K^\omega$ [15]. For example, in our simulation we choose $f^\omega$ to be B-spline (see supplementary mateiral). We focus on estimating the set of $f^\omega(x)$ with feature $x$ where the sparsity in the set specifies the graph structure.

We present the following lemma and theorem which show the equivalence between UGM and MVB:

**Lemma 2.1.** *In a MVB model, define the odd-even partition of the power set of $\omega$ as: $\Psi^\omega_{odd} = \{\kappa \subseteq \omega \mid |\kappa| = |\omega| - k, \text{where } k \text{ is odd}\}$, and $\Psi^\omega_{even} = \{\kappa \subseteq \omega \mid |\kappa| = |\omega| - k, \text{where } k \text{ is even}\}$. Note $|\Psi^\omega_{odd}| = |\Psi^\omega_{even}| = 2^{|\omega|-1}$. The following property holds:*

$$f^\omega = \log \frac{\prod_{\kappa \in \Psi^\omega_{even}} P(Y_i = 1, i \in \kappa; Y_j = 0, j \in \Omega \backslash \kappa | X)}{\prod_{\kappa \in \Psi^\omega_{odd}} P(Y_i = 1, i \in \kappa; Y_j = 0, j \in \Omega \backslash \kappa | X)}, \quad b(f) = \log \frac{Z(x)}{\prod_{C \in \mathcal{C}} \Phi_C(0; x)} \quad (3)$$

**Theorem 2.1.** *A UGM of the general form (1) with binary nodes is equivalent to a MVB model of (2). In addition, the following are equivalent: 1) There is no $|C|$-order interaction in $\{Y_i, i \in C\}$; 2) There is no clique $C \in \Psi_K$ in the graph; 3) $f^\omega = 0$ for all $\omega$ such that $C \subseteq \omega$.*

A proof is given in Appendix. It states that there is a clique $C$ in the graph, iff there is $\omega \supseteq C, f^\omega \neq 0$ in MVB model. The advantage of modeling by MVB is that the sparsity in $f^\omega$'s is sufficient and necessary for the conditional independence in the graph, thus fully specifying the graph structure. Specifically, $Y_i, Y_j$ are conditionally independent iff $f^\omega = 0, \omega \supseteq \{i, j\}$. This showed the interaction is non-zero iff all the nodes involved are not conditionally independent.

**Example 2.1.** *When $K = 2$, $\Omega = \{1, 2\}, \mathcal{C} = \{\Omega\}$, denote $\Phi_\Omega(Y_1 = 1, Y_2 = 1; X)$ as $\Phi_{11}$ for simplicity, then $P(Y_1 = 1, Y_2 = 1|X) = \frac{1}{Z} \Phi_{11}$. Define $\Phi_{10}, \Phi_{01}, \Phi_{00}$ similarly, then the distribution with UGM parameterization is determined. The relation between UGM and MVB is*

$$f^1 = \log \frac{\Phi_{10}}{\Phi_{00}}, \quad f^2 = \log \frac{\Phi_{01}}{\Phi_{00}}, \quad f^{1,2} = \log \frac{\Phi_{11} \cdot \Phi_{00}}{\Phi_{01} \cdot \Phi_{10}}$$

*Note, the independence between $Y_1$ and $Y_2$ implies: $f^{1,2} = 0$ or $\Phi_{11} \cdot \Phi_{00} = \Phi_{01} \cdot \Phi_{10}$. Therefore, $f^{1,2}$ being zero in MVB model is sufficient and necessary for the conditional independence in the model. On the other hand, $\log \Phi_C = 0$ is a sufficient condition but not necessary.*

The distribution of a general discrete UGM where $Y_k \in \{0, \cdots, m - 1\}$ can be extended from (2).

**Lemma 2.2.** *Let $V = \{1, \ldots, m - 1\}, y_\omega = (y_i)_{i \in \omega}$, then*

$$P(Y_1 = y_1, \cdots, Y_K = y_K | X) = \exp \Big\{ \sum_{\omega=1}^{\Omega} \sum_{v \in V^{|\omega|}} I(y_\omega = v) f^\omega_v - b(f) \Big\} \quad (4)$$

*where $I$ is an indicator function and $V^n$ is the tensor product of $n$ $V$'s. Each $f^\omega$ is a $|V|^{|\omega|}$ vector.*

## 3 Structure Penalty

In many applications, the assumption is that the graph has very few large cliques. Similar to the hierarchical inclusion restriction in Schmidt and Murphy [8], we will include a higher order interaction only when all its subsets are included. Our model is very flexible in that $f^\omega(x)$ can be in an arbitrary RKHS.

Let $y(i) = (y_1(i), \ldots, y_K(i)), x(i) = (x_1(i), \ldots, x_p(i))$ be the $i$th data point. There are $|\Psi_K| = 2^K - 1$ functions in total. We first consider learning the full model when $K$ is small, and later propose a greedy search algorithm to scale to large graphs. The penalized log likelihood model is:

$$\min I_\lambda(f) = L(f) + \lambda J(f) = \sum_{i=1}^{n} \Big( -\mathcal{Y}(i)^T f(x(i)) + b(f) \Big) + \lambda J(f) \quad (5)$$

where $L(f)$ is the negative log likelihood and $J(\cdot)$ is the structure penalty. The hierarchical assumption is that if there is no interaction on clique $C$, then all $f^\omega$ should be zero, for $\omega \supseteq C$. The penalty is designed to shrink such $f^\omega$ toward zero. We consider the Structure Lasso (SLasso) penalty guided by the lattice in Figure 2. The lattice $T$ has $2^K - 1$ nodes: $1, \ldots, \omega, \ldots, \Omega$. There is an edge from $\omega_1$ to $\omega_2$ if and only if $\omega_1 \subset \omega_2$ and $|\omega_1| + 1 = |\omega_2|$. Jenatton *et al.* [16] discussed how to define the groups to achieve different nonzero patterns.

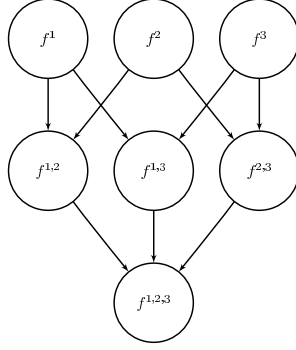

Figure 2: Hierarchical lattice for penalty

Let $T_v = \{\omega \in \Psi_K | v \subseteq \omega\}$ be the subgraph rooted at $v$ in $T$, including all the descendants of $v$. Denote $f^{T_v} = (f^\omega)_{\omega \in T_v}$. All the functions are categorized into groups with overlaps as $(T_1, \ldots, T_\Omega)$. The SLasso penalty on the group $T_v$ is: $J(f^{T_v}) = p_v \sqrt{\sum_{\omega \in T_v} \|f^\omega\|_{\mathcal{H}^\omega}^2}$ where $p_v$ is the weight for the penalty on $T_v$, empirically chosen as $\frac{1}{|T_v|}$. Then, the objective is:

$$\min_f \quad I_\lambda(f) = L(f) + \lambda \sum_v p_v \sqrt{\sum_{\omega \in T_v} \|f^\omega\|_{\mathcal{H}^\omega}^2} \qquad (6)$$

The following theorem shows that by minimizing the objective (6), $f^{\omega_1}$ will enter the model before $f^{\omega_2}$ if $\omega_1 \subset \omega_2$. That is to say, if $f^{\omega_1}$ is zero, there will be no higher order interactions on $\omega_2$. It is an extension of Theorem 1 in Zhao *et al.* [11] and the proof is given in Appendix.

**Theorem 3.1.** *Objective (6) is convex, thus the minimal is attainable. Let $\omega_1, \omega_2 \in \Psi_K$ and $\omega_1 \subset \omega_2$. If $\hat{f}$ is the minimizer of (6) given the observations, that is, $0 \in \partial I_\lambda(\hat{f})$ which is the subgradient of $I_\lambda$ at $\hat{f}$, then $\hat{f}^{\omega_2} = 0$ almost surely if $\hat{f}^{\omega_1} = 0$.*

**Example 3.1.** *If $K = 3$, $f = (f^1, f^2, f^3, f^{1,2}, f^{1,3}, f^{2,3}, f^{1,2,3})$. The group at node 1 in Figure 2 is $f^{T_1} = (f^1, f^{1,2}, f^{1,3}, f^{1,2,3})$ and $J(f^{T_1}) = p_1 \sqrt{\|f^1\|^2 + \|f^{1,2}\|^2 + \|f^{1,3}\|^2 + \|f^{1,2,3}\|^2}$.*

## 4   Parameter Estimation

In this section, we discuss parameter estimation where the $\omega$th function space is linear as $\mathcal{H}^\omega = \{1\} \oplus \mathcal{H}_1^\omega$ for simplicity. $\{1\}$ refers to the constant function space, and $\mathcal{H}_1^\omega$ is a RKHS with a linear kernel. The functions in $\mathcal{H}^\omega$ have the form $f^\omega(x) = c_0^\omega + \sum_{j=1}^p c_j^\omega x_j$. Its norm is $\|f^\omega\|_{\mathcal{H}^\omega} = \|c^\omega\|$, where $\|\cdot\|$ stands for Euclidean $l_2$ norm. Here, we denote $c^\omega = (c_0^\omega, \ldots, c_p^\omega)^T \in \mathbb{R}^{p+1}$ as a vector of length $p + 1$ and $c = (c^\omega)_{\omega \in \Psi_K} \in \mathbb{R}^{\tilde{p}}$ is the concatenated vector of all parameters of length $\tilde{p} = (p + 1) \cdot |\Psi_K|$. Let $c^{T_v} = (c^\omega)_{\omega \in T_v}$ be a $(p + 1) \cdot |T^v|$ vector, then the objective (6) is now:

$$\min_c \quad I_\lambda(c) = L(c) + \lambda \sum_v p_v \|c^{T_v}\| \qquad (7)$$

### 4.1   Estimating the complete model on small graphs

Many applications do not involve a large amount of responses, so it is desirable to learn the complete model when the graph is small for consistency reasons. We propose a method to optimize (7) of the

---
**Algorithm 1** Proximal Linearization Algorithm
---
> **Input:** $c_0, \alpha_0, \zeta > 1, tol > 0$
> **repeat**
>    Choose $\alpha_k \in [\alpha_{min}, \alpha_{max}]$
>    Solve Eq (8) for $d_k = c - c_k$
>    **while** $\delta_k = I_\lambda(c_k) - I_\lambda(c_k + d_k) < \|d_k\|^3$ **do**
>       // Insufficient decrease
>       Set $\alpha_k = \max(\alpha_{min}, \zeta\alpha_k)$
>       Solve Eq (8) for $d_k$
>    **end while**
>    Set $\alpha_{k+1} = \alpha_k/\zeta$
>    Set $c_{k+1} = c_k + d_k$
> **until** $\delta_k < tol$
---

complete model with all interaction levels by iteratively solving the following proximal linearization problem as discussed in Wright [12]:

$$\min_c L_k + \nabla L_k^T (c - c_k) + \frac{\alpha_k}{2}\|c - c_k\|^2 + \lambda J(c) \tag{8}$$

where $L_k = L(c_k)$, and $\alpha_k$ is a positive scalar chosen adaptively at $k$th step. With slight abuse of notation, we denote $c_k$ as the value of $c$ at $k$th step. Algorithm 1 summarized the framework of solving (7). Following the analysis in Wright [12], we can ensure that the proximal linearization algorithm will converge for the negative log-likelihood loss function with the SLasso penalty.

However, solving group lasso with overlaps is not trivial due to the non-smoothness at the singular point. In recent years, several papers have addressed this problem. Jacob *et al.* [10] duplicated the design matrix columns that appear in group overlaps, then solved the problem as group lasso without overlaps. Kim and Xing [17] reparameterized the group norm with additional dummy variables. They alternatively optimized the model parameters and the dummy ones at each step. It is efficient for the quadratic loss function on Gaussian data, but might not scale well in our case. Instead, we solve (8) by its smooth and convex dual problem [18].The details are in the supplementary material.

## 4.2 Estimating large graphs

The above algorithm is efficient on small graphs ($K < 20$). It usually terminates within 20 iterations in our experiments. However, the issue of estimating a complete model is the exponential number of $f^\omega$'s and the same amount of groups involved in objective (7). It is intractable when the graph becomes large. The hierarchical assumption and the SLasso penalty lend themselves naturally to a greedy search algorithm:

1. Start from the set of main effects as $A_0 = \{f^1, \cdots, f^K\}$.
2. In step $i$, remove the nodes that are not in $A_i$ from the lattice in Figure 2. Obtain a sparse estimation of the functions in $A_i$ by algorithm (1). Denote the resulting sparse set $A_i'$.
3. Let $A_{i+1} = A_i'$. Keep adding a higher order interaction into $A_{i+1}$ if all its subsets of interactions are included in $A_i'$. And also add this node into the lattice in Figure 2.

Iterate step 2 and 3 until convergence. The algorithm is similar to the active set method in Schmidt and Murphy [8]. It has multiple runs of algorithm (1) to enforce the hierarchical assumption. It is not guaranteed to converge to the global optimum. Nonetheless, our empirical experiments show its ability to scale to large graphs.

# 5 Experiments

## 5.1 Toy Data

In the simulation, we create 6 toy graphs. The first four graphs are depicted in Figure 1. Graph 5 has 100 nodes where the first 8 nodes have the same structure as in Figure 1(c) and the others are independent. Graph 6 also has 100 nodes where the first 10 nodes have the same connection as in Figure 1(d) and the others are independent. We generate 100 datasets for each structure to evaluate

the performance. The sample size of each dataset is 1000. Here is how the first data set is generated: The length of the feature vector, $p$, is set to 5 in our experiment, i.e. $X = (X_1, \ldots, X_5)$. Each $f^\omega(x) = c_0^\omega + \sum_{j=1}^5 g_j^\omega(x_j)$ where $g_j^\omega(x_j) = \sum_{k=1}^D c_{jk}^\omega B_k(x_j)$ is spanned by the B-spline basis functions $\{B_k(\cdot)\}_{k=1,\cdots,D}$ (see the supplementary material), where $D$ is chosen to be 5. The true set of the model parameters, $c_{jk}^\omega$, is uniformly sampled from $\{-5, -4, \cdots, 5\}$. We set the intercepts $c_0^\omega$ in main effects to 1, and those in second or higher order interactions to 2. The features, $X_j$, are i.i.d uniform on [-1, 1]. Then, $Y$ is sampled according to the probability in equation (2).

We use GACV (generalized approximate cross validation) and BGACV (B-type GACV) [19] to choose the regularization parameter $\lambda$ for the complete model (graphs 1-4). We call these variants of SLasso Complete-GACV and Complete-BGACV. We use AIC for greedy search (Greedy-AIC) in graphs 5 and 6 due to computational consideration. The range of $\lambda$ is chosen according to Koh *et al.* [20]. The details of the tuning methods are discussed in the supplementary material. The R package, BMN, is used as a baseline [4].

Table 1: Number of true positive and false positive functions

| Graph | Method | $f^{1,2}$ | $f^{1,3}$ | $f^{2,3}$ | $f^{3,4}$ | $f^{1,2,3}$ | $f^{5,7,8}$ | $f^{5,6,7,8}$ | FP |
|---|---|---|---|---|---|---|---|---|---|
| 1 | BMN | 60 | 76 | 70 | 60 | 0 | - | - | 162 |
| | Complete-GACV | 100 | 100 | 100 | 94 | 84 | - | - | 136 |
| | Complete-BGACV | 86 | 83 | 83 | 72 | 14 | - | - | 11 |
| 2 | BMN | 44 | 50 | 38 | 58 | 0 | - | - | 412 |
| | Complete-GACV | 100 | 99 | 100 | 99 | 83 | - | - | 341 |
| | Complete-BGACV | 88 | 91 | 88 | 78 | 33 | - | - | 64 |
| 3 | BMN | 72 | 64 | 60 | 60 | 0 | 0 | 0 | 830 |
| | Complete-GACV | 91 | 87 | 81 | 92 | 62 | 71 | 33 | 412 |
| | Complete-BGACV | 36 | 22 | 23 | 93 | 0 | 39 | 0 | 162 |
| 4 | BMN | 48 | 34 | 37 | 29 | 0 | 0 | - | 774 |
| | Complete-GACV | 92 | 98 | 94 | 90 | 54 | 45 | - | 693 |
| | Complete-BGACV | 68 | 68 | 71 | 62 | 0 | 0 | - | 144 |
| 5 | BMN | 38 | 28 | 26 | 22 | 0 | 0 | 0 | 9476 |
| | Greedy-AIC | 99 | 99 | 98 | 97 | 22 | 21 | 0 | 1997 |
| 6 | BMN | 28 | 26 | 14 | 26 | 0 | 0 | - | 9672 |
| | Greedy-AIC | 100 | 100 | 100 | 99 | 24 | 15 | - | 3458 |

In Table 1, we count, for each function $f^\omega$, the number of runs out of 100 where $f^\omega$ is recovered ($\|c^\omega\| \neq 0$). If a recovered function is in the true model, it is considered a true positive, otherwise a false positive. The main effects are always detected correctly, thus are not listed in the table. SLasso is more effective compared to BMN which only considers pairwise interactions.

In Figure 3, we show the learning results in terms of true positive rate (TPR) as sample size increases from 100 to 1000. The experimental setting is the same as before. The TPRs improve with increasing sample size. GACV achieves better TPR, but higher FPR compared to BGACV. Our method outperforms BMN in all six graphs.

## 5.2 Case Study: Census Bureau County Data

We use the county data from U.S. Census Bureau[1] to validate our method. We remove the counties that have missing values and obtain 2668 entries in total. The outcomes of this study are summarized in Table 2. "Vote" [21] is coded as 1 if the Republican candidate won in the 2004 presidential election. To dichotomize the remaining outcomes, the national mean is selected as a threshold. The data is standardized to mean 0 and variance 1. The following features are included: Housing unit change in percent from 2000-2006, percent of ethnic groups, percent foreign born, percent people over 65, percent people under 18, percent people with a high school education, percent people with a bachelors degree; birth rate, death rate, per capita government expenditure in dollars. By adjusting $\lambda$, we observe new interactions enter the model. The graph structure of $\lambda = 0.1559$ is

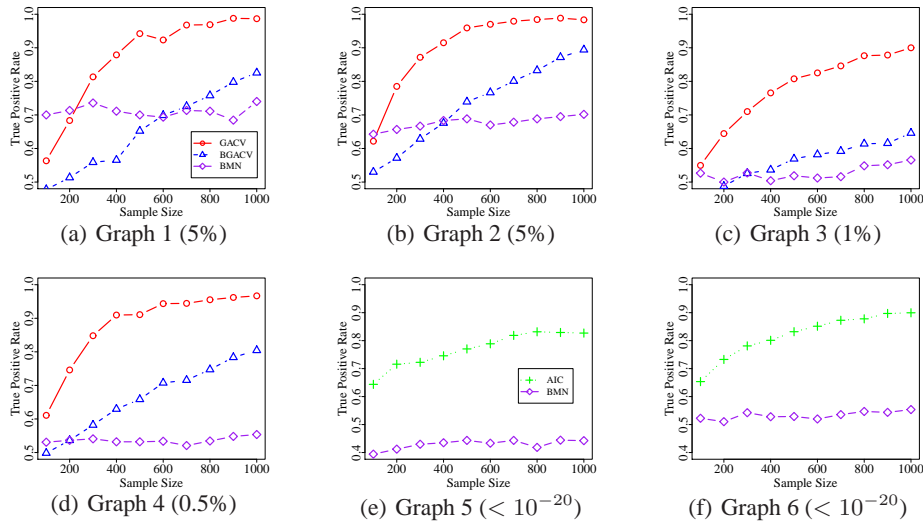

(a) Graph 1 (5%)     (b) Graph 2 (5%)     (c) Graph 3 (1%)

(d) Graph 4 (0.5%)     (e) Graph 5 ($< 10^{-20}$)     (f) Graph 6 ($< 10^{-20}$)

Figure 3: The True Positive Rate (TPR) of graph structure learning methods with increasing sample size. The percentage in the bracket is the upper bound of False Positive Rate (FPR) in each experiment. BMN always has larger FPR compared to SLasso.

Table 2: Selected response variables

| Response | Description | Positive% |
|---|---|---|
| Vote | 2004 votes for Republican presidential candidate | 81.11 |
| Poverty | Poverty Rate | 52.70 |
| VCrime | Violent Crime Rate, eg. murder, robbery | 23.09 |
| PCrime | Property Crime Rate, eg. burglary | 6.82 |
| URate | Unemployment Rate | 51.35 |
| PChange | Population change in percent from 2000 to 2006 | 64.96 |

shown in Figure 4(a). The results of BMN (the tuning parameter is 0.015) is in Figure 4(b). The unemployment rate plays an important role as a hub as discovered by SLasso, but not by BMN.

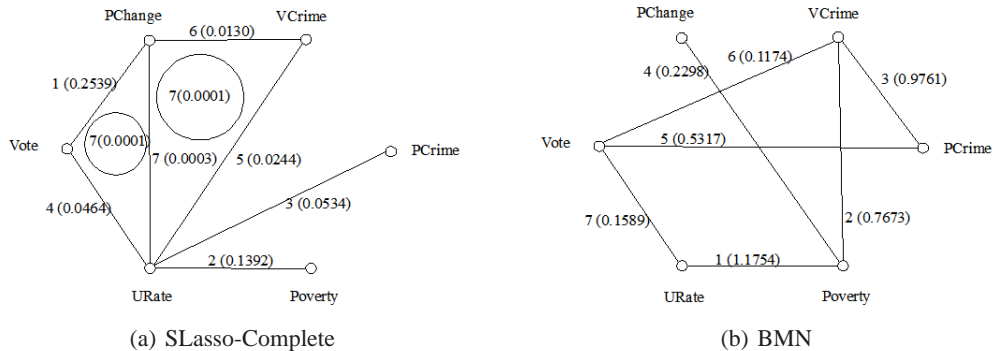

(a) SLasso-Complete     (b) BMN

Figure 4: Interactions of response variables in the Census Bureau data. The first number on the edge is the order at which the link is recovered. The number in bracket is the function norm on the clique and the absolute value of the elements in the concentration matrix, respectively. We note SLasso discovers at 7th step two third-order interactions which are displayed by two circles in (a).

We analyze the link between "Vote" and "PChange". Though the marginal correlation between them (without $X$) is only 0.0389, which is the second lowest absolute pairwise correlation, the

link is firstly recovered by SLasso. It has been suggested that there is indeed a connection[2]. This shows that after taking features into account, the dependence structure of response variables may change and hidden relations could be discovered. The main factors in this case are "percentage of housing unit change" ($X_1$) and "population percentage of people over 65" ($X_2$). The part of the fitted model shown below suggests that as housing units increase, the counties are more likely to have both positive results for "Vote" and "PChange". But this tendency will be counteracted by the increase of people over 65: the responses are less likely to take both positive values.

$$\hat{f}^{Vote} = 0.2913 \cdot X_1 + 0.3475 \cdot X_2 + \cdots$$
$$\hat{f}^{PChange} = 1.4726 \cdot X_1 - 0.3709 \cdot X_2 + \cdots$$
$$\hat{f}^{Vote,PChange} = 0.1358 \cdot X_1 - 0.0458 \cdot X_2 + \cdots$$

## 6   Conclusions

Our SLasso method can learn the graph structure that is specified by the conditional log odds ratios conditioned on input features $X$, which allows the graphical model depending on features. The modeling interprets well, since $f^\omega = 0$ iff there is no such clique. An efficient algorithm is given to estimate the complete model. A greedy approach is applied when the graph is large. SLasso can be extended to model a general discrete UGM, where $Y_k$ takes value in $\{0, \ldots, m-1\}$. Also, there exist rich selections of the function forms, which makes the model more flexible and powerful, though modification is needed in solving the proximal subproblem for non-parametric families.

## A   Proof

### A.1   Proof of Theorem 2.1

*Proof.* Given UGM (1), the corresponding parameterization in MVB model is shown in (3) of Lemma 2.1. Conversely, given the MVB model of (2), the cliques can be determined by the nonzero $f^\omega$: clique $C$ exists if $C = \omega$ and $f^\omega \neq 0$. Then the maximal cliques can be inferred from the graph structure. And suppose they are $C_1, \ldots, C_m$. Let $\omega_i = C_i$, for $i = 1, \ldots, m$, and $\kappa_1 = \emptyset$, $\kappa_i = C_i \cap (C_{i-1} \cup \cdots \cup C_1), i = 2, \ldots, m$. Then the parameterization is:

$$\Phi_{C_i}(y_{C_i}; x) = \exp\left(S^{\omega_i}(y; x) - S^{\kappa_i}(y; x)\right) \quad \text{and} \quad Z(x) = \exp(b(f)) \tag{9}$$

where $S^\omega(y; x) = \sum_{\kappa \subseteq \omega} y^\kappa f^\kappa(x)$. Thus, UGM (1) with bivariate nodes is equivalent to MVB (2).

In the latter part of the theorem, $1 \Rightarrow 2$ and $3 \Rightarrow 1$ follow naturally from the Markov property of graphical models. To show $2 \Rightarrow 3$, let $y_C^\omega$ be a realization of $y_C$ such that $y_C^\omega = (y_i^\omega)_{i \in C}$ where $y_i^\omega = 1$ if $i \in \omega$ and $y_i^\omega = 0$ otherwise. Notice that whenever $\kappa \cap C = \kappa' \cap C$, we have $y_C^\kappa = y_C^{\kappa'}$. For any possible $v = \kappa \cap C, \kappa' \in \{\kappa | \kappa = v \cup u, \text{ s.t. } u \subseteq \omega - v\}$ will satisfy the condition: $\kappa' \cap C = v$. There are $2^{|\omega - v|}$ such $\kappa'$ in total due to the choice of $u$. Also, they appear in the nominator and denominator of equation (3) equally. So, for any $C \in \mathcal{C}$,

$$\prod_{\kappa \in \Psi_{even}^\omega} \Phi_C(y_C^\kappa; x) = \prod_{\kappa \in \Psi_{odd}^\omega} \Phi_C(y_C^\kappa; x) \tag{10}$$

It follows that $f^\omega = 0$ by (3). □

### A.2   Proof of Theorem 3.1

*Proof.* We give the proof for the linear case. The convexity of $I_\lambda$ is easy to check, since $L$ and $J(f^{T_v})$ are all convex in $c$. Suppose there is some $\omega_2 \supset \omega_1$ s.t. $\hat{c}^{\omega_2} \neq 0$ and $\hat{c}^{\omega_1} = 0$, by the groups constructed through Figure 2, $\|\hat{c}^{T_v}\| = \|(\hat{c}^\omega)_{v \subseteq \omega}\| \neq 0$ for all $v \subseteq \omega_1$. So the partial derivative of the objective (7) with respect to $c^{\omega_1}$ at $\hat{c}^{\omega_1}$ is

$$\left.\frac{\partial L}{\partial c^{\omega_1}}\right|_{c^{\omega_1} = \hat{c}^{\omega_1}} + \lambda \sum_{v \subseteq \omega_1} p_v \frac{\hat{c}^{\omega_1}}{\|\hat{c}^{T_v}\|} = 0 \tag{11}$$

Thus, the probability of $\{\hat{c}^{\omega_2} \neq 0\}$ equals to the probability of $\{\left.\frac{\partial L}{\partial c^{\omega_1}}\right|_{c^{\omega_1} = \hat{c}^{\omega_1}} = 0\}$, which is 0. □

## Footnotes

*SD wishes to acknowledge the valuable comments from Stephen J. Wright and Sijian Wang. Research of SD and GW is supported in part by NIH Grant EY09946, NSF Grant DMS-0906818 and ONR Grant N0014-09-1-0655. Research of XZ is supported in part by NSF IIS-0953219, IIS-0916038.

[1]http://www.census.gov/statab/www/ccdb.html

[2] http://www.ipsos-mori.com/researchpublications/researcharchive/2545/Analysis-Population-change-turnout-the-election.aspx

## References

[1] N. Meinshausen and P. Buhlmann. High-dimensional graphs and variable selection with the lasso. *The Annals of Statistics*, 34(3):1436–1462, 2006.

[2] J. Peng, P. Wang, N. Zhou, and J. Zhu. Partial correlation estimation by joint sparse regression models. *Journal of the American Statistical Association*, 104(486):735–746, 2009.

[3] P. Ravikumar, M.J. Wainwright, and J. Lafferty. High-dimensional Ising model selection using l1-regularized logistic regression. *Annals of Statistics*, 38(3):1287–1319, 2010.

[4] H. Höfling and R. Tibshirani. Estimation of sparse binary pairwise markov networks using pseudo-likelihoods. *The Journal of Machine Learning Research*, 10:883–906, 2009.

[5] Han Liu, Xi Chen, John Lafferty, and Larry Wasserman. Graph-valued regression. In J. Lafferty, C. K. I. Williams, J. Shawe-Taylor, R.S. Zemel, and A. Culotta, editors, *Advances in Neural Information Processing Systems 23*, pages 1423–1431. 2010.

[6] M. Schmidt, K. Murphy, G. Fung, and R. Rosales. Structure learning in random fields for heart motion abnormality detection. In *IEEE Conference on Computer Vision and Pattern Recognition*, pages 1–8, 2008.

[7] J.K. Bradley and C. Guestrin. Learning tree conditional random fields. In *Proceedings of the 27th International Conference on Machine learning*, pages 127–134, 2010.

[8] M. Schmidt and K. Murphy. Convex structure learning in log-linear models: Beyond pairwise potentials. In *Proceedings of the International Conference on Artificial Intelligence and Statistics (AISTATS)*, 2010.

[9] M. Yuan and Y. Lin. Model selection and estimation in regression with grouped variables. *Journal of the Royal Statistical Society: Series B (Statistical Methodology)*, 68(1):49–67, 2006.

[10] L. Jacob, G. Obozinski, and J.P. Vert. Group Lasso with overlap and graph Lasso. In *Proceedings of the 26th Annual International Conference on Machine Learning*, pages 433–440, 2009.

[11] P. Zhao, G. Rocha, and B. Yu. The composite absolute penalties family for grouped and hierarchical variable selection. *Annals of Statistics*, 37(6A):3468–3497, 2009.

[12] S.J. Wright. Accelerated block-coordinate relaxation for regularized optimization. Technical report, Department of Computer Science, University of Wisconsin-Madison, 2010.

[13] M.J. Wainwright and M.I. Jordan. Graphical models, exponential families, and variational inference. *Foundations and Trends® in Machine Learning*, 1:1–305, 2008.

[14] F. Gao, G. Wahba, R. Klein, and B. Klein. Smoothing Spline ANOVA for multivariate Bernoulli observations, with application to ophthalmology data. *Journal of the American Statistical Association*, 96(453):127, 2001.

[15] G. Wahba. *Spline Models for Observational Data*. Society for Industrial Mathematics, 1990.

[16] R. Jenatton, J.Y. Audibert, and F. Bach. Structured variable selection with sparsity-inducing norms. *arXiv:0904.3523*, 2009.

[17] S. Kim and E.P. Xing. Tree-guided group lasso for multi-task regression with structured sparsity. In *Proceedings of 27th International Conference on Machine Learning*, pages 543–550, Haifa, Israel, 2010.

[18] J. Liu and J. Ye. Fast overlapping group lasso. *arXiv:1009.0306v1*, 2010.

[19] Xiwen Ma. *Penalized Regression in Reproducing Kernel Hilbert Spaces With Randomized Covariate Data*. PhD thesis, Department of Statistics, University of Wisconsin-Madison, 2010.

[20] K. Koh, S.J. Kim, and S. Boyd. An interior-point method for large-scale l1-regularized logistic regression. *Journal of Machine learning research*, 8(8):1519–1555, 2007.

[21] R.M. Scammon, A.V. McGillivray, and R. Cook. *America Votes 26: 2003-2004, Election Returns By State*. CQ Press, 2005.

